# Efficient Sampling for Gaussian Process Inference using Control Variables

**Michalis K. Titsias, Neil D. Lawrence and Magnus Rattray**
School of Computer Science, University of Manchester
Manchester M13 9PL, UK

## Abstract

Sampling functions in Gaussian process (GP) models is challenging because of the highly correlated posterior distribution. We describe an efficient Markov chain Monte Carlo algorithm for sampling from the posterior process of the GP model. This algorithm uses control variables which are auxiliary function values that provide a low dimensional representation of the function. At each iteration, the algorithm proposes new values for the control variables and generates the function from the conditional GP prior. The control variable input locations are found by minimizing an objective function. We demonstrate the algorithm on regression and classification problems and we use it to estimate the parameters of a differential equation model of gene regulation.

## 1   Introduction

Gaussian processes (GPs) are used for Bayesian non-parametric estimation of unobserved or latent functions. In regression problems with Gaussian likelihoods, inference in GP models is analytically tractable, while for classification deterministic approximate inference algorithms are widely used [16, 4, 5, 11]. However, in recent applications of GP models in systems biology [1] that require the estimation of ordinary differential equation models [2, 13, 8], the development of deterministic approximations is difficult since the likelihood can be highly complex. Other applications of Gaussian processes where inference is intractable arise in spatio-temporal models and geostatistics and deterministic approximations have also been developed there [14]. In this paper, we consider Markov chain Monte Carlo (MCMC) algorithms for inference in GP models. An advantage of MCMC over deterministic approximate inference is that it provides an arbitrarily precise approximation to the posterior distribution in the limit of long runs. Another advantage is that the sampling scheme will often not depend on details of the likelihood function, and is therefore very generally applicable.

In order to benefit from the advantages of MCMC it is necessary to develop an efficient sampling strategy. This has proved to be particularly difficult in many GP applications, because the posterior distribution describes a highly correlated high-dimensional variable. Thus simple MCMC sampling schemes such as Gibbs sampling can be very inefficient. In this contribution we describe an efficient MCMC algorithm for sampling from the posterior process of a GP model which constructs the proposal distributions by utilizing the GP prior. This algorithm uses control variables which are auxiliary function values. At each iteration, the algorithm proposes new values for the control variables and samples the function by drawing from the conditional GP prior. The control variables are highly informative points that provide a low dimensional representation of the function. The control input locations are found by minimizing an objective function. The objective function used is the expected least squares error of reconstructing the function values from the control variables, where the expectation is over the GP prior.

We demonstrate the proposed MCMC algorithm on regression and classification problems and compare it with two Gibbs sampling schemes. We also apply the algorithm to inference in a systems

biology model where a set of genes is regulated by a transcription factor protein [8]. This provides an example of a problem with a non-linear and non-factorized likelihood function.

## 2   Sampling algorithms for Gaussian Process models

In a GP model we assume a set of inputs $(\mathbf{x}_1, \ldots, \mathbf{x}_N)$ and a set of function values $\mathbf{f} = (f_1, \ldots, f_N)$ evaluated at those inputs. A Gaussian process places a prior on $\mathbf{f}$ which is a $N$-dimensional Gaussian distribution so that $p(\mathbf{f}) = N(\mathbf{y}|\boldsymbol{\mu}, K)$. The mean $\boldsymbol{\mu}$ is typically zero and the covariance matrix $K$ is defined by the kernel function $k(\mathbf{x}_n, \mathbf{x}_m)$ that depends on parameters $\boldsymbol{\theta}$. GPs are widely used for supervised learning [11] in which case we have a set of observed pairs $(\mathbf{y}_i, \mathbf{x}_i)$, where $i = 1, \ldots, N$, and we assume a likelihood model $p(\mathbf{y}|\mathbf{f})$ that depends on parameters $\boldsymbol{\alpha}$. For regression or classification problems, the latent function values are evaluated at the observed inputs and the likelihood factorizes according to $p(\mathbf{y}|\mathbf{f}) = \prod_{i=1}^{N} p(y_i|f_i)$. However, for other type of applications, such as modelling latent functions in ordinary differential equations, the above factorization is not applicable. Assuming that we have obtained suitable values for the model parameters $(\boldsymbol{\theta}, \boldsymbol{\alpha})$ inference over $\mathbf{f}$ is done by applying Bayes rule:

$$p(\mathbf{f}|\mathbf{y}) \propto p(\mathbf{y}|\mathbf{f})p(\mathbf{f}). \tag{1}$$

For regression, where the likelihood is Gaussian, the above posterior is a Gaussian distribution that can be obtained using simple algebra. When the likelihood $p(\mathbf{y}|\mathbf{f})$ is non-Gaussian, computations become intractable and we need to carry out approximate inference.

The MCMC algorithm we consider is the general Metropolis-Hastings (MH) algorithm [12]. Suppose we wish to sample from the posterior in eq. (1). The MH algorithm forms a Markov chain. We initialize $\mathbf{f}^{(0)}$ and we consider a proposal distribution $Q(\mathbf{f}^{(t+1)}|\mathbf{f}^{(t)})$ that allows us to draw a new state given the current state. The new state is accepted with probability $\min(1, A)$ where

$$A = \frac{p(\mathbf{y}|\mathbf{f}^{(t+1)})p(\mathbf{f}^{(t+1)})}{p(\mathbf{y}|\mathbf{f}^{(t)})p(\mathbf{f}^{(t)})} \frac{Q(\mathbf{f}^{(t)}|\mathbf{f}^{(t+1)})}{Q(\mathbf{f}^{(t+1)}|\mathbf{f}^{(t)})}. \tag{2}$$

To apply this generic algorithm, we need to choose the proposal distribution $Q$. For GP models, finding a good proposal distribution is challenging since $\mathbf{f}$ is high dimensional and the posterior distribution can be highly correlated.

To motivate the algorithm presented in section 2.1, we discuss two extreme options for specifying the proposal distribution $Q$. One simple way to choose $Q$ is to set it equal to the GP prior $p(\mathbf{f})$. This gives us an independent MH algorithm [12]. However, sampling from the GP prior is very inefficient as it is unlikely to obtain a sample that will fit the data. Thus the Markov chain will get stuck in the same state for thousands of iterations. On the other hand, sampling from the prior is appealing because any generated sample satisfies the smoothness requirement imposed by the covariance function. Functions drawn from the posterior GP process should satisfy the same smoothness requirement as well.

The other extreme choice for the proposal, that has been considered in [10], is to apply Gibbs sampling where we iteratively draw samples from each posterior conditional density $p(f_i|\mathbf{f}_{-i}, \mathbf{y})$ with $\mathbf{f}_{-i} = \mathbf{f} \setminus f_i$. However, Gibbs sampling can be extremely slow for densely discretized functions, as in the regression problem of Figure 1, where the posterior GP process is highly correlated. To clarify this, note that the variance of the posterior conditional $p(f_i|\mathbf{f}_{-i}, \mathbf{y})$ is smaller or equal to the variance of the conditional GP prior $p(f_i|\mathbf{f}_{-i})$. However, $p(f_i|\mathbf{f}_{-i})$ may already have a tiny variance caused by the conditioning on all remaining latent function values. For the one-dimensional example in Figure 1, Gibbs sampling is practically not applicable. We further study this issue in section 4.

A similar algorithm to Gibbs sampling can be expressed by using the sequence of the conditional densities $p(f_i|\mathbf{f}_{-i})$ as a proposal distribution for the MH algorithm[1]. We call this algorithm the Gibbs-like algorithm. This algorithm can exhibit a high acceptance rate, but it is inefficient to sample from highly correlated functions. A simple generalization of the Gibbs-like algorithm that is more appropriate for sampling from smooth functions is to divide the domain of the function into regions and sample the entire function within each region by conditioning on the remaining function regions. Local region sampling iteratively draws each block of functions values $\mathbf{f}_k$ from

the conditional GP prior $p(\mathbf{f}_k^{t+1}|\mathbf{f}_{-k}^{(t)})$, where $\mathbf{f}_{-k} = \mathbf{f} \setminus \mathbf{f}_k$. However, this scheme is still inefficient to sample from highly correlated functions since the variance of the proposal distribution can be very small close to the boundaries between neighbouring function regions. The description of this algorithm is given in the supplementary material. In the next section we discuss an algorithm using control variables that can efficiently sample from highly correlated functions.

## 2.1 Sampling using control variables

Let $\mathbf{f}_c$ be a set of $M$ auxiliary function values that are evaluated at inputs $X_c$ and drawn from the GP prior. We call $\mathbf{f}_c$ the control variables and their meaning is analogous to the auxiliary inducing variables used in sparse GP models [15]. To compute the posterior $p(\mathbf{f}|\mathbf{y})$ based on control variables we use the expression

$$p(\mathbf{f}|\mathbf{y}) = \int_{\mathbf{f}_c} p(\mathbf{f}|\mathbf{f}_c, \mathbf{y}) p(\mathbf{f}_c|\mathbf{y}) d\mathbf{f}_c. \tag{3}$$

Assuming that $\mathbf{f}_c$ is highly informative about $\mathbf{f}$, so that $p(\mathbf{f}|\mathbf{f}_c, \mathbf{y}) \simeq p(\mathbf{f}|\mathbf{f}_c)$, we can approximately sample from $p(\mathbf{f}|\mathbf{y})$ in a two-stage manner: firstly sample the control variables from $p(\mathbf{f}_c|\mathbf{y})$ and then generate $\mathbf{f}$ from the conditional prior $p(\mathbf{f}|\mathbf{f}_c)$. This scheme can allow us to introduce a MH algorithm, where we need to specify only a proposal distribution $q(\mathbf{f}_c^{(t+1)}|\mathbf{f}_c^{(t)})$, that will mimic sampling from $p(\mathbf{f}_c|\mathbf{y})$, and always sample $\mathbf{f}$ from the conditional prior $p(\mathbf{f}|\mathbf{f}_c)$. The whole proposal distribution takes the form

$$Q(\mathbf{f}^{(t+1)}, \mathbf{f}_c^{(t+1)}|\mathbf{f}^{(t)}, \mathbf{f}_c^{(t)}) = p(\mathbf{f}^{(t+1)}|\mathbf{f}_c^{(t+1)}) q(\mathbf{f}_c^{(t+1)}|\mathbf{f}_c^{(t)}). \tag{4}$$

Each proposed sample is accepted with probability $\min(1, A)$ where $A$ is given by

$$A = \frac{p(\mathbf{y}|\mathbf{f}^{(t+1)}) p(\mathbf{f}_c^{(t+1)})}{p(\mathbf{y}|\mathbf{f}^{(t)}) p(\mathbf{f}_c^{(t)})} \cdot \frac{q(\mathbf{f}_c^{(t)}|\mathbf{f}_c^{(t+1)})}{q(\mathbf{f}_c^{(t+1)}|\mathbf{f}_c^{(t)})}. \tag{5}$$

The usefulness of the above sampling scheme stems from the fact that the control variables can form a low-dimensional representation of the function. Assuming that these variables are much fewer than the points in $\mathbf{f}$, the sampling is mainly carried out in the low dimensional space. In section 2.2 we describe how to select the number $M$ of control variables and the inputs $X_c$ so as $\mathbf{f}_c$ becomes highly informative about $\mathbf{f}$. In the remainder of this section we discuss how we set the proposal distribution $q(\mathbf{f}_c^{(t+1)}|\mathbf{f}_c^{(t)})$.

A suitable choice for $q$ is to use a Gaussian distribution with diagonal or full covariance matrix. The covariance matrix can be adapted during the burn-in phase of MCMC in order to increase the acceptance rate. Although this scheme is general, it has practical limitations. Firstly, tuning a full covariance matrix is time consuming and in our case this adaption process must be carried out simultaneously with searching for an appropriate set of control variables. Also, since the terms involving $p(\mathbf{f}_c)$ do not cancel out in the acceptance probability in eq. (5), using a diagonal covariance for the $q$ distribution has the risk of proposing control variables that may not satisfy the GP prior smoothness requirement. To avoid these problems, we define $q$ by utilizing the GP prior. According to eq. (3) a suitable choice for $q$ must mimic the sampling from the posterior $p(\mathbf{f}_c|\mathbf{y})$. Given that the control points are far apart from each other, Gibbs sampling in the control variables space can be efficient. However, iteratively sampling $f_{c_i}$ from the conditional posterior $p(f_{c_i}|\mathbf{f}_{c_{-i}}, \mathbf{y}) \propto p(\mathbf{y}|\mathbf{f}_c) p(f_{c_i}|\mathbf{f}_{c_{-i}})$, where $\mathbf{f}_{c_{-i}} = \mathbf{f}_c \setminus f_{c_i}$ is intractable for non-Gaussian likelihoods[2]. An attractive alternative is to use a Gibbs-like algorithm where each $f_{c_i}$ is drawn from the conditional GP prior $p(\mathbf{f}_{c_i}^{(t+1)}|\mathbf{f}_{c_{-i}}^{(t)})$ and is accepted using the MH step. More specifically, the proposal distribution draws a new $f_{c_i}^{(t+1)}$ for a certain control variable $i$ from $p(f_{c_i}^{(t+1)}|\mathbf{f}_{c_{-i}}^{(t)})$ and generates the function $\mathbf{f}^{(t+1)}$ from $p(\mathbf{f}^{(t+1)}|f_{c_i}^{(t+1)}, \mathbf{f}_{c_{-i}}^{(t)})$. The sample $(f_{c_i}^{(t+1)}, \mathbf{f}^{(t+1)})$ is accepted using the MH step. This scheme of sampling the control variables one-at-a-time and resampling $\mathbf{f}$ is iterated between different control variables. A complete iteration of the algorithm consists of a full scan over all control variables. The acceptance probability $A$ in eq. (5) becomes the likelihood ratio and the prior smoothness requirement is always satisfied. The iteration between different control variables is illustrated in Figure 1.

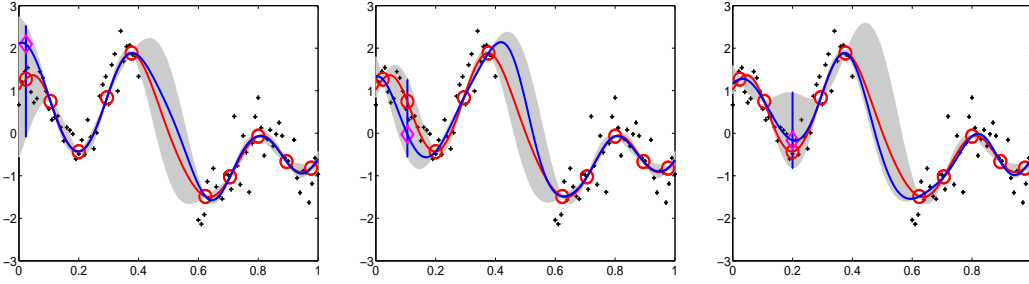

Figure 1: Visualization of iterating between control variables. The red solid line is the current $\mathbf{f}^{(t)}$, the blue line is the proposed $\mathbf{f}^{(t+1)}$, the red circles are the current control variables $\mathbf{f}_c^{(t)}$ while the diamond (in magenta) is the proposed control variable $f_{c_i}^{(t+1)}$. The blue solid vertical line represents the distribution $p(f_{c_i}^{(t+1)}|\mathbf{f}_{c-i}^{(t)})$ (with two-standard error bars) and the shaded area shows the effective proposal $p(\mathbf{f}^{t+1}|\mathbf{f}_{c-i}^{(t)})$.

Although the control variables are sampled one-at-at-time, $\mathbf{f}$ can still be drawn with a considerable variance. To clarify this, note that when the control variable $f_{c_i}$ changes the effective proposal distribution for $\mathbf{f}$ is $p(\mathbf{f}^{t+1}|\mathbf{f}_{c-i}^{(t)}) = \int_{f_{c_i}^{(t+1)}} p(\mathbf{f}^{t+1}|f_{c_i}^{(t+1)}, \mathbf{f}_{c-i}^{(t)}) p(f_{c_i}^{(t+1)}|\mathbf{f}_{c-i}^{(t)}) df_{c_i}^{(t+1)}$, which is the conditional GP prior given all the control points apart from the current point $f_{c_i}$. This conditional prior can have considerable variance close to $f_{c_i}$ and in all regions that are not close to the remaining control variables. As illustrated in Figure 1, the iteration over different control variables allow $\mathbf{f}$ to be drawn with a considerable variance everywhere in the input space.

## 2.2   Selection of the control variables

To apply the previous algorithm we need to select the number, $M$, of the control points and the associated inputs $X_c$. $X_c$ must be chosen so that knowledge of $\mathbf{f}_c$ can determine $\mathbf{f}$ with small error. The prediction of $\mathbf{f}$ given $\mathbf{f}_c$ is equal to $K_{f,c}K_{c,c}^{-1}\mathbf{f}_c$ which is the mean of the conditional prior $p(\mathbf{f}|\mathbf{f}_c)$. A suitable way to search over $X_c$ is to minimize the reconstruction error $||\mathbf{f} - K_{f,c}K_{c,c}^{-1}\mathbf{f}_c||^2$ averaged over any possible value of $(\mathbf{f}, \mathbf{f}_c)$:

$$G(X_c) = \int_{\mathbf{f}, \mathbf{f}_c} ||\mathbf{f} - K_{f,c}K_{c,c}^{-1}\mathbf{f}_c||^2 p(\mathbf{f}|\mathbf{f}_c)p(\mathbf{f}_c) d\mathbf{f} d\mathbf{f}_c = \text{Tr}(K_{f,f} - K_{f,c}K_{c,c}^{-1}K_{f,c}^T).$$

The quantity inside the trace is the covariance of $p(\mathbf{f}|\mathbf{f}_c)$ and thus $G(X_c)$ is the total variance of this distribution. We can minimize $G(X_c)$ w.r.t. $X_c$ using continuous optimization similarly to the approach in [15]. Note that when $G(X_c)$ becomes zero, $p(\mathbf{f}|\mathbf{f}_c)$ becomes a delta function.

To find the number $M$ of control points we minimize $G(X_c)$ by incrementally adding control variables until the total variance of $p(\mathbf{f}|\mathbf{f}_c)$ becomes smaller than a certain percentage of the total variance of the prior $p(\mathbf{f})$. $5\%$ was the threshold used in all our experiments. Then we start the simulation and we observe the acceptance rate of the Markov chain. According to standard heuristics [12] which suggest that desirable acceptance rates of MH algorithms are around $1/4$, we require a full iteration of the algorithm (a complete scan over the control variables) to have an acceptance rate larger than $1/4$. When for the current set of control inputs $X_c$ the chain has a low acceptance rate, it means that the variance of $p(\mathbf{f}|\mathbf{f}_c)$ is still too high and we need to add more control points in order to further reduce $G(X_c)$. The process of observing the acceptance rate and adding control variables is continued until we reach the desirable acceptance rate.

When the training inputs $X$ are placed uniformly in the space, and the kernel function is stationary, the minimization of $G$ places $X_c$ in a regular grid. In general, the minimization of $G$ places the control inputs close to the clusters of the input data in such a way that the kernel function is taken into account. This suggests that $G$ can also be used for learning inducing variables in sparse GP models in a unsupervised fashion, where the observed outputs $\mathbf{y}$ are not involved.

# 3   Applications

We consider two applications where exact inference is intractable due to a non-linear likelihood function: classification and parameter estimation in a differential equation model of gene regulation.

**Classification**: Deterministic inference methods for GP classification are described in [16, 4, 7]. Among these approaches, the Expectation-Propagation (EP) algorithm [9] is found to be the most efficient [6]. Our MCMC implementation confirms these findings since sampling using control variables gave similar classification accuracy to EP.

**Transcriptional regulation**: We consider a small biological sub-system where a set of target genes are regulated by one transcription factor (TF) protein. Ordinary differential equations (ODEs) can provide an useful framework for modelling the dynamics in these biological networks [1, 2, 13, 8]. The concentration of the TF and the gene specific kinetic parameters are typically unknown and need to be estimated by making use of a set of observed gene expression levels. We use a GP prior to model the unobserved TF activity, as proposed in [8], and apply full Bayesian inference based on the MCMC algorithm presented previously.

Barenco et al. [2] introduce a linear ODE model for gene activation from TF. This approach was extended in [13, 8] to account for non-linear models. The general form of the ODE model for transcription regulation with a single TF has the form

$$\frac{dy_j(t)}{dt} = B_j + S_j g(f(t)) - D_j y_j(t), \tag{6}$$

where the changing level of a gene $j$'s expression, $y_j(t)$, is given by a combination of basal transcription rate, $B_j$, sensitivity, $S_j$, to its governing TF's activity, $f(t)$, and the decay rate of the mRNA, $D_j$. The differential equation can be solved for $y_j(t)$ giving

$$y_j(t) = \frac{B_j}{D_j} + A_j e^{-D_j t} + S_j e^{-D_j t} \int_0^t g(f(u)) e^{D_j u} du, \tag{7}$$

where $A_j$ term arises from the initial condition. Due to the non-linearity of the $g$ function that transforms the TF, the integral in the above expression is not analytically obtained. However, numerical integration can be used to accurately approximate the integral with a dense grid $(u_i)_{i=1}^P$ of points in the time axis and evaluating the function at the grid points $f_p = f(u_p)$. In this case the integral in the above equation can be written $\sum_{p=1}^{P_t} w_p g(f_p) e^{D_j u_p}$ where the weights $w_p$ arise from the numerical integration method used and, for example, can be given by the composite Simpson rule.

The TF concentration $f(t)$ in the above system of ODEs is a latent function that needs to be estimated. Additionally, the kinetic parameters of each gene $\boldsymbol{\alpha}_j = (B_j, D_j, S_j, A_j)$ are unknown and also need to be estimated. To infer these quantities we use mRNA measurements (obtained from microarray experiments) of $N$ target genes at $T$ different time steps. Let $y_{jt}$ denote the observed gene expression level of gene $j$ at time $t$ and let $\mathbf{y} = \{y_{jt}\}$ collect together all these observations. Assuming a Gaussian noise for the observed gene expressions the likelihood of our data has the form

$$p(\mathbf{y}|\mathbf{f}, \{\boldsymbol{\alpha}_j\}_{j=1}^N) = \prod_{j=1}^N \prod_{t=1}^T p(y_{jt}|\mathbf{f}_{1 \leq p \leq P_t}, \boldsymbol{\alpha}_j), \tag{8}$$

where each probability density in the above product is a Gaussian with mean given by eq. (7) and $\mathbf{f}_{1 \leq p \leq P_t}$ denotes the TF values up to time $t$. Notice that this likelihood is non-Gaussian due to the non-linearity of $g$. Further, this likelihood does not have a factorized form, as in the regression and classification cases, since an observed gene expression depends on the protein concentration activity in all previous times points. Also note that the discretization of the TF in $P$ time points corresponds to a very dense grid, while the gene expression measurements are sparse, i.e. $P \gg T$.

To apply full Bayesian inference in the above model, we need to define prior distributions over all unknown quantities. The protein concentration $\mathbf{f}$ is a positive quantity, thus a suitable prior is to consider a GP prior for $\log \mathbf{f}$. The kinetic parameters of each gene are all positive scalars. Those parameters are given vague gamma priors. Sampling the GP function is done exactly as described in section 2; we have only to plug in the likelihood from eq. (8) in the MH step. Sampling from the kinetic parameters is carried using Gaussian proposal distributions with diagonal covariance matrices that sample the positive kinetic parameters in the log space.

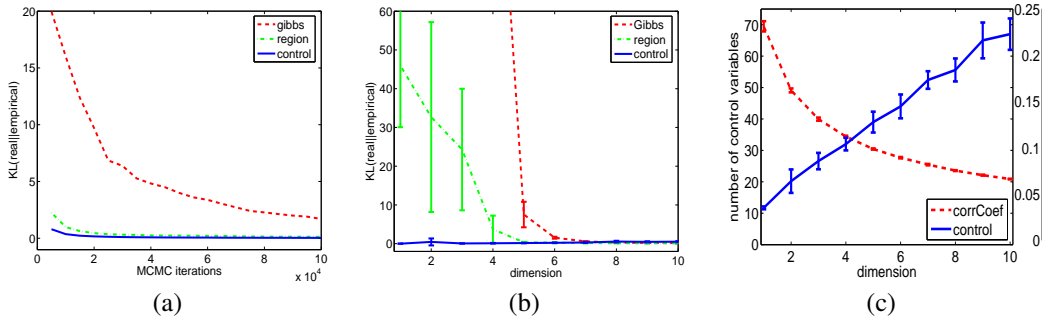

Figure 2: (a) shows the evolution of the KL divergence (against the number of MCMC iterations) between the true posterior and the empirically estimated posteriors for a 5-dimensional regression dataset. (b) shows the mean values with one-standard error bars of the KL divergence (against the input dimension) between the true posterior and the empirically estimated posteriors. (c) plots the number of control variables together with the average correlation coefficient of the GP prior.

## 4    Experiments

In the first experiment we compare Gibbs sampling (*Gibbs*), sampling using local regions (*region*) (see the supplementary file) and sampling using control variables (*control*) in standard regression problems of varied input dimensions. The performance of the algorithms can be accurately assessed by computing the KL divergences between the exact Gaussian posterior $p(\mathbf{f}|\mathbf{y})$ and the Gaussians obtained by MCMC. We fix the number of training points to $N = 200$ and we vary the input dimension $d$ from 1 to 10. The training inputs $X$ were chosen randomly inside the unit hypercube $[0,1]^d$. Thus, we can study the behavior of the algorithms w.r.t. the amount of correlation in the posterior GP process which depends on how densely the function is sampled. The larger the dimension, the sparser the function is sampled. The outputs $Y$ were chosen by randomly producing a GP function using the squared-exponential kernel $\sigma_f^2 \exp(-\frac{||\mathbf{x}_m - \mathbf{x}_n||^2}{2\ell^2})$, where $(\sigma_f^2, \ell^2) = (1, 100)$ and then adding noise with variance $\sigma^2 = 0.09$. The burn-in period was $10^4$ iterations[3]. For a certain dimension $d$ the algorithms were initialized to the same state obtained by randomly drawing from the GP prior. The parameters $(\sigma_f^2, \ell^2, \sigma^2)$ were fixed to the values that generated the data. The experimental setup was repeated 10 times so as to obtain confidence intervals. We used thinned samples (by keeping one sample every 10 iterations) to calculate the means and covariances of the 200-dimensional posterior Gaussians. Figure 2(a) shows the KL divergence against the number of MCMC iterations for the 5-dimensional input dataset. It seems that for 200 training points and 5 dimensions, the function values are still highly correlated and thus *Gibbs* takes much longer for the KL divergence to drop to zero. Figure 2(b) shows the KL divergence against the input dimension after fixing the number of iterations to be $3 \times 10^4$. Clearly *Gibbs* is very inefficient in low dimensions because of the highly correlated posterior. As dimension increases and the functions become sparsely sampled, *Gibbs* improves and eventually the KL divergences approaches zero. The *region* algorithm works better than *Gibbs* but in low dimensions it also suffers from the problem of high correlation. For the *control* algorithm we observe that the KL divergence is very close to zero for all dimensions. Figure 2(c) shows the increase in the number of control variables used as the input dimension increases. The same plot shows the decrease of the average correlation coefficient of the GP prior as the input dimension increases. This is very intuitive, since one should expect the number of control variables to increase as the function values become more independent.

Next we consider two GP classification problems for which exact inference is intractable. We used the Wisconsin Breast Cancer (WBC) and the Pima Indians Diabetes (PID) binary classification datasets. The first consists of 683 examples (9 input dimensions) and the second of 768 examples (8 dimensions). 20% of the examples were used for testing in each case. The MCMC samplers were run for $5 \times 10^4$ iterations (thinned to one sample every five iterations) after a burn-in of $10^4$ iterations. The hyperparameters were fixed to those obtained by EP. Figures 3(a) and (b) shows

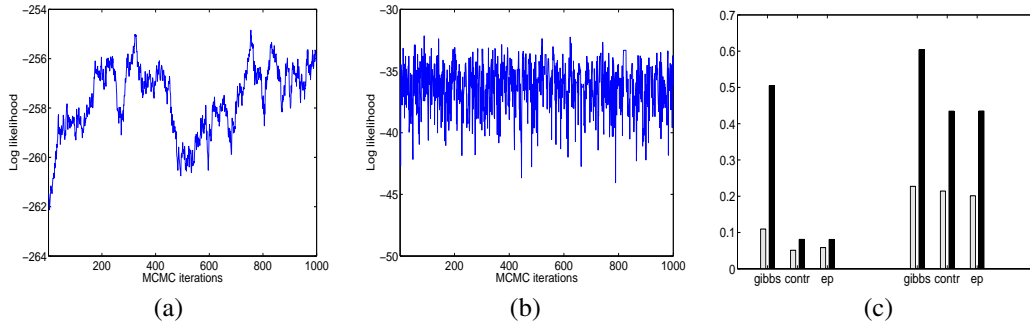

(a)                                    (b)                                    (c)

Figure 3: We show results for GP classification. Log-likelihood values are shown for MCMC samples obtained from (a) *Gibbs* and (b) *control* applied to the WBC dataset. In (c) we show the test errors (grey bars) and the average negative log likelihoods (black bars) on the WBC (left) and PID (right) datasets and compare with EP.

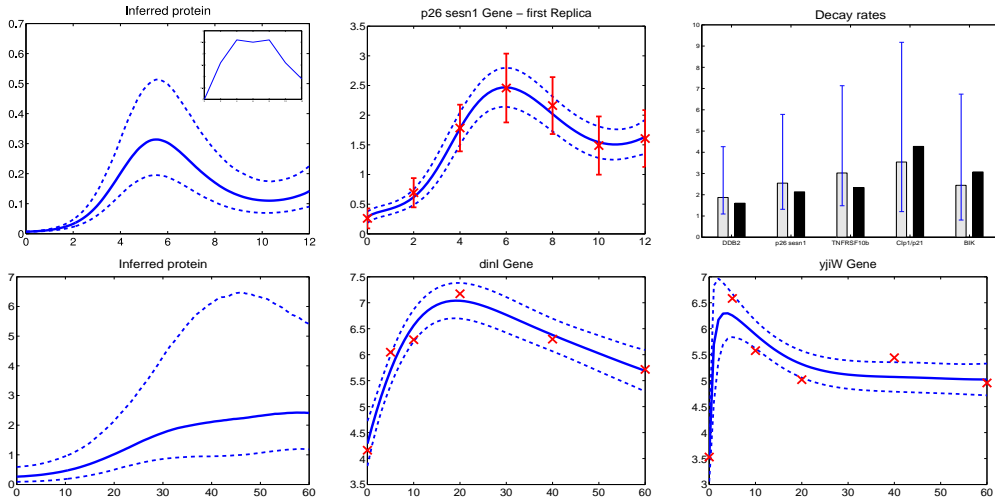

Figure 4: **First row:** The left plot shows the inferred TF concentration for p53; the small plot on top-right shows the ground-truth protein concentration obtained by a *Western blot* experiment [2]. The middle plot shows the predicted expression of a gene obtained by the estimated ODE model; red crosses correspond to the actual gene expression measurements. The right-hand plot shows the estimated decay rates for all 5 target genes used to train the model. Grey bars display the parameters found by MCMC and black bars the parameters found in [2] using a linear ODE model. **Second row**: The left plot shows the inferred TF for LexA. Predicted expressions of two target genes are shown in the rest two plots. Error bars in all plots correspond to $95\%$ credibility intervals.

the log-likelihood for MCMC samples on the WBC dataset, for the *Gibbs* and *control* algorithms respectively. It can be observed that mixing is far superior for the *control* algorithm and it has also converged to a much higher likelihood. In Figure 3(c) we compare the test error and the average negative log likelihood in the test data obtained by the two MCMC algorithms with the results from EP. The proposed *control* algorithm shows similar classification performance to EP, while the Gibbs algorithm performs significantly worse on both datasets.

In the final two experiments we apply the *control* algorithm to infer the protein concentration of TFs that activate or repress a set of target genes. The latent function in these problems is always one-dimensional and densely discretized and thus the *control* algorithm is the only one that can converge to the GP posterior process in a reasonable time.

We first consider the TF p53 which is a tumour repressor activated during DNA damage. Seven samples of the expression levels of five target genes in three replicas are collected as the raw time course data. The non-linear activation of the protein follows the Michaelis Menten kinetics inspired response [1] that allows saturation effects to be taken into account so as $g(f(t)) = \frac{f(t)}{\gamma_j + f(t)}$ in eq.

(6) where the Michaelis constant for the jth gene is given by $\gamma_j$. Note that since $f(t)$ is positive the GP prior is placed on the $\log f(t)$. To apply MCMC we discretize $\mathbf{f}$ using a grid of $P = 121$ points. During sampling, 7 control variables were needed to obtain the desirable acceptance rate. Running time was 4 hours for $5 \times 10^5$ sampling iterations plus $5 \times 10^4$ burn-in iterations. The first row of Figure 4 summarizes the estimated quantities obtained from MCMC simulation.

Next we consider the TF LexA in E.Coli that acts as a repressor. In the repression case there is an analogous Michaelis Menten model [1] where the non-linear function $g$ takes the form: $g(f(t)) = \frac{1}{\gamma_j + f(t)}$. Again the GP prior is placed on the log of the TF activity. We applied our method to the same microarray data considered in [13] where mRNA measurements of 14 target genes are collected over six time points. For this dataset, the expression of the 14 genes were available for $T = 6$ times. The GP function $\mathbf{f}$ was discretized using 121 points. The result for the inferred TF profile along with predictions of two target genes are shown in the second row of Figure 4. Our inferred TF profile and reconstructed target gene profiles are similar to those obtained in [13]. However, for certain genes, our model provides a better fit to the gene profile.

## 5 Discussion

Gaussian processes allow for inference over latent functions using a Bayesian estimation framework. In this paper, we presented an MCMC algorithm that uses control variables. We showed that this sampling scheme can efficiently deal with highly correlated posterior GP processes. MCMC allows for full Bayesian inference in the transcription factor networks application. An important direction for future research will be scaling the models used to much larger systems of ODEs with multiple interacting transcription factors. In such large systems where MCMC can become slow a combination of our method with the fast sampling scheme in [3] could be used to speed up the inference.

**Acknowledgments**

This work is funded by EPSRC Grant No EP/F005687/1 "Gaussian Processes for Systems Identification with Applications in Systems Biology".

## Footnotes

[1]Thus we replace the proposal distribution $p(f_i|\mathbf{f}_{-i}, \mathbf{y})$ with the prior conditional $p(f_i|\mathbf{f}_{-i})$.

[2]This is because we need to integrate out $\mathbf{f}$ in order to compute $p(\mathbf{y}|\mathbf{f}_c)$.

[3]For *Gibbs* we used $2 \times 10^4$ iterations since the *region* and *control* algorithms require additional iterations during the adaption phase.

## References

[1] U. Alon. *An Introduction to Systems Biology: Design Principles of Biological Circuits*. Chapman and Hall/CRC, 2006.

[2] M. Barenco, D. Tomescu, D. Brewer, J. Callard, R. Stark, and M. Hubank. Ranked prediction of p53 targets using hidden variable dynamic modeling. *Genome Biology*, 7(3), 2006.

[3] B. Calderhead, M. Girolami, and N.D. Lawrence. Accelerating Bayesian Inference over Nonlinear Differential Equations with Gaussian Processes. In *Neural Information Processing Systems, 22*, 2008.

[4] L. Csato and M. Opper. Sparse online Gaussian processes. *Neural Computation*, 14:641–668, 2002.

[5] M. N. Gibbs and D. J. C. MacKay. Variational Gaussian process classifiers. *IEEE Transactions on Neural Networks*, 11(6):1458–1464, 2000.

[6] M. Kuss and C. E. Rasmussen. Assessing Approximate Inference for Binary Gaussian Process Classification. *Journal of Machine Learning Research*, 6:1679–1704, 2005.

[7] N. D. Lawerence, M. Seeger, and R. Herbrich. Fast sparse Gaussian process methods: the informative vector machine. In *Advances in Neural Information Processing Systems, 13*. MIT Press, 2002.

[8] N. D. Lawrence, G. Sanguinetti, and M. Rattray. Modelling transcriptional regulation using Gaussian processes. In *Advances in Neural Information Processing Systems, 19*. MIT Press, 2007.

[9] T. Minka. Expectation propagation for approximate Bayesian inference. In *UAI*, pages 362–369, 2001.

[10] R. M. Neal. Monte Carlo implementation of Gaussian process models for Bayesian regression and classification. Technical report, Dept. of Statistics, University of Toronto, 1997.

[11] C. E. Rasmussen and C. K. I. Williams. *Gaussian Processes for Machine Learning*. MIT Press, 2006.

[12] C. P. Robert and G. Casella. *Monte Carlo Statistical Methods*. Springer-Verlag, 2nd edition, 2004.

[13] S. Rogers, R. Khanin, and M. Girolami. Bayesian model-based inference of transcription factor activity. *BMC Bioinformatics*, 8(2), 2006.

[14] H. Rue, S. Martino, and N. Chopin. Approximate Bayesian inference for latent Gaussian models using integrated nested Laplace approximations. *NTNU Statistics Preprint*, 2007.

[15] E. Snelson and Z. Ghahramani. Sparse Gaussian process using pseudo inputs. In *Advances in Neural Information Processing Systems, 13*. MIT Press, 2006.

[16] C. K. I. Williams and D. Barber. Bayesian classification with Gaussian processes. *IEEE Transactions on Pattern Analysis and Machine Intelligence*, 20(12):1342–1351, 1998.
